# MCMC for continuous-time discrete-state systems

**Vinayak Rao**
Gatsby Computational Neuroscience Unit
University College London
vrao@gatsby.ucl.ac.uk

**Yee Whye Teh**
Gatsby Computational Neuroscience Unit
University College London
ywteh@gatsby.ucl.ac.uk

## Abstract

We propose a simple and novel framework for MCMC inference in continuous-time discrete-state systems with pure jump trajectories. We construct an *exact* MCMC sampler for such systems by alternately sampling a random discretization of time given a trajectory of the system, and then a new trajectory given the discretization. The first step can be performed efficiently using properties of the Poisson process, while the second step can avail of discrete-time MCMC techniques based on the forward-backward algorithm. We show the advantage of our approach compared to particle MCMC and a uniformization-based sampler.

## 1 Introduction

There has been growing interest in the machine learning community to model dynamical systems in continuous time. Examples include point processes [1], Markov processes [2], structured Markov processes [3], infinite state Markov processes [4], semi-Markov processes [5] etc. However, a major impediment towards the more widespread use of these models is the problem of inference. A simple approach is to discretize time, and then run inference on the resulting approximation. This however has a number of drawbacks, not least of which is that we lose the advantages that motivated the use of continuous time in the first place. Time-discretization introduces a bias into our inferences, and to control this, one has to work at a time resolution that results in a very large number of discrete time steps. This can be computationally expensive.

Our focus in this paper is on posterior sampling via Markov chain Monte Carlo (MCMC), and there is a huge literature on such techniques for *discrete-time* models [6]. Here, we construct an *exact* MCMC sampler for pure jump processes in continuous time, using a workhorse of the discrete-time domain, the forward-filtering backward-sampling algorithm [7, 8], to make efficient updates.

The core of our approach is an auxiliary variable Gibbs sampler that repeats two steps. The first step runs the forward-backward algorithm on a *random* discretization of time to sample a new trajectory. The second step then resamples a new time-discretization given this trajectory. A random discretization allows a relatively coarse grid, while still keeping inferences unbiased. Such a coarse discretization allows us to apply the forward-backward algorithm to a Markov chain with relatively few time steps, resulting in computational savings. Even though the *marginal* distribution of the random time-discretization can be quite complicated, we show that *conditioned* on the system trajectory, it is just distributed as a Poisson process.

While the forward-backward algorithm was developed originally for finite state hidden Markov models and linear Gaussian systems, it also forms the core of samplers for more complicated systems like nonlinear/non-Gaussian [9], infinite state [10], and non-Markovian [11] time series. Our ideas thus apply to essentially any pure jump process, so long as it makes only finite transitions over finite intervals. For concreteness, we focus on semi-Markov processes. We compare our sampler with two other continuous-time MCMC samplers, a particle MCMC sampler [12], and a uniformization-based sampler [13]. The latter turns out to be a special case of ours, corresponding to a random time-discretization that is marginally distributed as a homogeneous Poisson process.

## 2 Semi-Markov processes

A semi-Markov (jump) process (sMJP) is a right-continuous, piecewise-constant stochastic process on the nonnegative real-line taking values in some state space $\mathcal{S}$ [14, 15]. For simplicity, we assume $\mathcal{S}$ is finite, labelling its elements from $1$ to $N$. We also assume the process is stationary. Then, the sMJP is parametrized by $\pi_0$, an (arbitrary) initial distribution over states, as well as an $N \times N$ matrix of *hazard functions*, $A_{ss'}(\cdot)$ $\forall s, s' \in \mathcal{S}$. For any $\tau$, $A_{ss'}(\tau)$ gives the rate of transitioning to state $s'$, $\tau$ time units after entering state $s$ (we allow self-transitions, so $s'$ can equal $s$). Let this transition occur after a waiting time $\tau_{s'}$. Then $\tau_{s'}$ is distributed according to the density $r_{ss'}(\cdot)$, related to $A_{ss'}(\cdot)$ as shown below (see eg. [16]):

$$r_{ss'}(\tau_{s'}) = A_{ss'}(\tau_{s'})e^{\left(-\int_0^{\tau_{s'}} A_{ss'}(u)\mathrm{d}u\right)}, \quad A_{ss'}(\tau_{s'}) = r_{ss'}(\tau_{s'})/\left(1 - \int_0^{\tau_{s'}} r_{ss'}(u)\mathrm{d}u\right) \quad (1)$$

Sampling an sMJP trajectory proceeds as follows: on entering state $s$, sample waiting times $\tau_{s'} \sim A_{ss'}(\cdot)$ $\forall s' \in \mathcal{S}$. The sMJP enters a new state, $s_{new}$, corresponding to the smallest of these waiting times. Let this waiting time be $\tau_{hold}$ (so that $\tau_{hold} = \tau_{s_{new}} = \min_{s'} \tau_{s'}$). Then, advance the current time by $\tau_{hold}$, and set the sMJP state to $s_{new}$. Repeat this procedure, now with the rate functions $A_{s_{new}s'}(\cdot)$ $\forall s' \in \mathcal{S}$.

Define $A_s(\cdot) = \sum_{s' \in \mathcal{S}} A_{ss'}(\cdot)$. From the independence of the times $\tau_{ss'}$, equation 1 tells us that

$$P(\tau_{hold} > \tau) = \prod_{s' \in \mathcal{S}} P(\tau_{s'} > \tau) = e^{\left(-\int_0^{\tau} A_s(u)\mathrm{d}u\right)}, \quad \tau_{hold} \sim r_s(\tau) \equiv A_s(\tau)e^{\left(-\int_0^{\tau} A_s(u)\mathrm{d}u\right)} \quad (2)$$

Comparing with equation 1, we see that $A_s(\cdot)$ gives the rate of *any* transition out of state $s$. An equivalent characterization of many continuous-time processes is to first sample the waiting time $\tau_{hold}$, and then draw a new state $s'$. For the sMJP, the latter probability is proportional to $A_{ss'}(\tau_{hold})$.

A special sMJP is the Markov jump process (MJP) where the hazard functions are constant (giving exponential waiting times). For an MJP, future behaviour is independent of the current waiting time. By allowing general waiting-time distributions, an sMJP can model memory effects like burstiness or refractoriness in the system dynamics.

We represent an sMJP trajectory on an interval $[t_{start}, t_{end}]$ as $(S, T)$, where $T = (t_0, \cdots, t_{|T|})$ is the sequence of jump times (including the endpoints) and $S = (s_0, \cdots, s_{|S|})$ is the corresponding sequence of state values. Here $|S| = |T|$, and $s_{i+1} = s_i$ implies a self-transition at time $t_{i+1}$ (except at the end time $t_{|T|} = t_{end}$ which does not correspond to a jump). The filled circles in figure 1(c) represent $(S, T)$; since the process is right-continuous, $s_i$ gives the state *after* the jump at $t_i$.

### 2.1 Sampling by dependent thinning

We now describe an alternate thinning-based approach to sampling an sMJP trajectory. Our approach will produce *candidate* event times at a rate higher that the actual event rates in the system. To correct for this, we probabilistically reject (or thin) these events. Define $W$ as the sequence of actual event times $T$, together with the thinned event times (which we call $U$, these are the empty circles in figure 1(c)). $W = (w_0, \cdots, w_{|W|})$ forms a random discretization of time (with $|W| = |T| + |U|$); define $V = (v_0, \cdots, v_{|W|})$ as a sequence of state assignments to the times $W$. At any $w_i$, let $l_i$ represent the time since the last sMJP transition (so that, $l_i = w_i - \max_{t \in T, t \leq w_i} t$), and let $L = (l_1, \cdots, l_{|W|})$. Figures 1(b) and (c) show these quantities, as well as continuous-time processes $\mathbf{S}(t)$ and $\mathbf{L}(t)$ such that $l_i = \mathbf{L}(w_i)$ and $s_i = \mathbf{S}(w_i)$. $(V, L, W)$ forms an equivalent representation of $(S, T)$ that includes a redundant set of thinned events $U$. Note that if the $i$th event is thinned, $v_i = v_{i-1}$, however this is *not* a self-transition. $L$ helps distinguish self-transitions (having associated $l$'s equal to 0) from thinned events. We explain the generative process of $(V, L, W)$ below; a proof of its correctness is included in the supplementary material.

For each hazard function $A_s(\tau)$, define another dominating hazard function $B_s(\tau)$, so that $B_s(\tau) \geq A_s(\tau)$ $\forall s, \tau$. Suppose we have instantiated the system trajectory until time $w_i$, with the sMJP having just entered state $v_i \in \mathcal{S}$ (so that $l_i = 0$). We sample the next candidate event time $w_{i+1}$, with $\Delta w_i = (w_{i+1} - w_i)$ drawn from the hazard function $B_{v_i}(\cdot)$. A larger rate implies faster events, so that $\Delta w_i$ will on average be smaller than a waiting time $\tau_{hold}$ drawn from $A_{v_i}(\cdot)$. We correct for this by treating $w_{i+1}$ as an actual event with probability $\frac{A_{v_i}(\Delta w_i + l_i)}{B_{v_i}(\Delta w_i + l_i)}$. If this is the case, we sample a new state $v_{i+1}$ with probability proportional to $A_{v_i v_{i+1}}(\Delta w_i + l_i)$, and set $l_{i+1} = 0$. On the other hand, if the event is rejected, we set $v_{i+1}$ to $v_i$, and $l_{i+1} = (\Delta w_i + l_i)$. We now sample

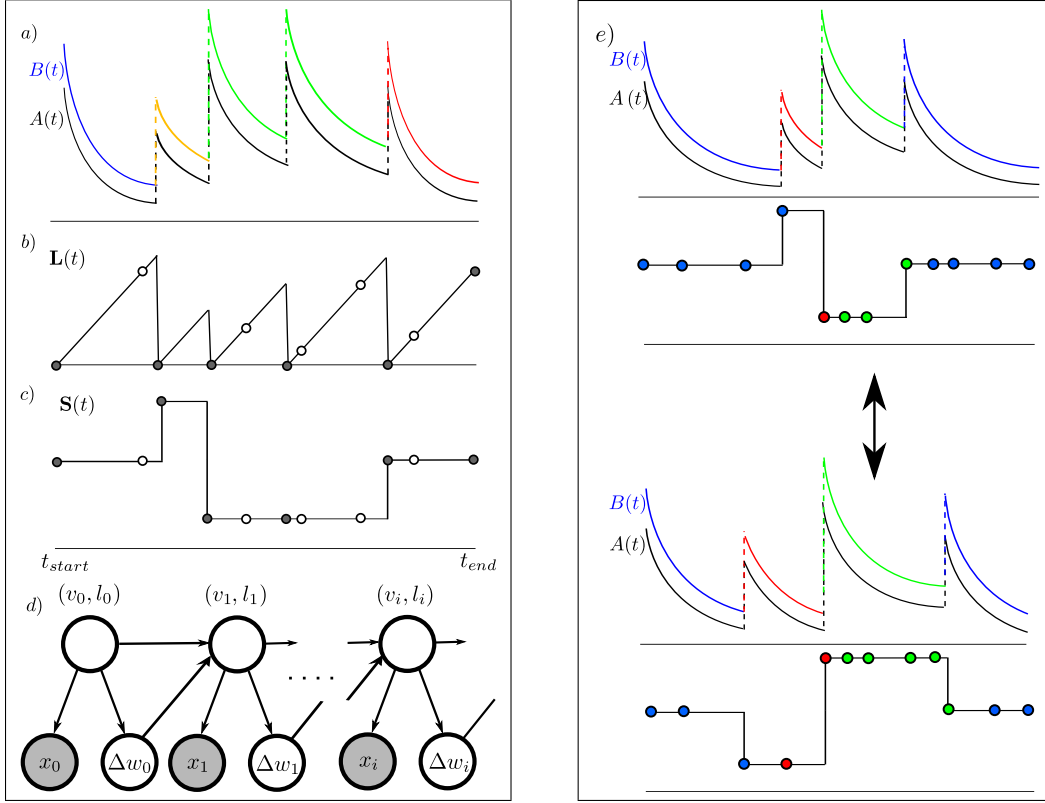

Figure 1: a) Instantaneous hazard rates given a trajectory b) State holding times, $\mathbf{L}(t)$ c) sMJP state values $\mathbf{S}(t)$ d) Graphical model for the randomized time-discretization e) Resampling the sMJP trajectory. In b) and c), the filled and empty circles represent actual and thinned events respectively.

$\Delta w_{i+1}$ (and thus $w_{i+2}$), such that $(\Delta w_{i+1} + l_{i+1}) \sim B_{v_{i+1}}(\cdot)$. More simply, we sample a new waiting time from $B_{v_{i+1}}(\cdot)$, *conditioned on it being greater than $l_{i+1}$*. Again, accept this point with probability $\frac{A_{v_{i+1}}(\Delta w_{i+1}+l_{i+1})}{B_{v_{i+1}}(\Delta w_{i+1}l_{i+1})}$, and repeat this process. Proposition 1 confirms that this generative process (summarized by the graphical model in figure 1(d), and algorithm 1) yields a trajectory from the sMJP. Figure 1(d) also depicts observations $X$ of the sMJP trajectory; we elaborate on this later.

**Proposition 1.** *The path $(V, L, W)$ returned by the thinning procedure described above is equivalent to a sample $(S, T)$ from the sMJP $(\pi_0, A)$.*

---

**Algorithm 1** State-dependent thinning for sMJPs

---

Input:     Hazard functions $A_{ss'}(\cdot) \; \forall s, s' \in \mathcal{S}$, and an initial distribution over states $\pi_0$.
            Dominating hazard functions $B_s(\tau) \geq A_s(\tau) \; \forall \tau, s$, where $A_s(\tau) = \sum_{s'} A_{ss'}(\tau)$.
Output:    A piecewise constant path $(V, L, W) \equiv ((v_i, l_i, w_i))$ on the interval $[t_{start}, t_{end}]$.

1: Draw $v_0 \sim \pi_0$ and set $w_0 = t_{start}$. Set $l_0 = 0$ and $i = 0$.
2: **while** $w_i < t_{end}$ **do**
3:     Sample $\tau_{hold} \sim B_{v_i}(\cdot)$, with $\tau_{hold} > l_i$. Let $\Delta w_i = \tau_{hold} - l_i$, and $w_{i+1} = w_i + \Delta w_i$.
4:         **with probability** $\frac{A_{v_i}(\tau_{hold})}{B_{v_i}(\tau_{hold})}$
5:             Set $l_{i+1} = 0$, and sample $v_{i+1}$, with $P(v_{i+1} = s'|v_i) \propto A_{v_i s'}(\tau_{hold}), \; s' \in \mathcal{S}$.
6:         **else**
7:             Set $l_{i+1} = l_i + \Delta w_i$, and $v_{i+1} = v_i$.
8:         **end**
9:     Increment $i$.
10: **end while**
11: Set $w_{|W|} = t_{end}, v_{|W|} = v_{|W|-1}, l_{|W|} = l_{|W|} + w_{|W|} - w_{|W|-1}$.

---

## 2.2 Posterior inference via MCMC

We now define an auxiliary variable Gibbs sampler, setting up a Markov chain that converges to the posterior distribution over the thinned representation $(V, L, W)$ given observations $X$ of the sMJP trajectory. The observations can lie in any space $\mathcal{X}$, and for any time-discretization $W$, let $x_i$ represent all observations in the interval $(w_i, w_{i+1})$. By construction, the sMJP stays in a single state $v_i$ over this interval; let $P(x_i|v_i)$ be the corresponding likelihood vector. Given a time discretization $W \equiv (U \cup T)$ and the observations $X$, we discard the old state labels $(V, L)$, and sample a new path $(\tilde{V}, \tilde{L}, W) \equiv (\tilde{S}, \tilde{T})$ using the forward-backward algorithm. We then discard the thinned events $\tilde{U}$, and given the path $(\tilde{S}, \tilde{T})$, resample new thinned events $U_{new}$, resulting in a new time discretization $W_{new} \equiv (\tilde{T} \cup U_{new})$. We describe both operations below.

**Resampling the sMJP trajectory given the set of times $W$:**
Given $W$ (and thus all $\Delta w_i$), this involves assigning each element $w_i \in W$, a label $(v_i, l_i)$ (see figure 1(d)). Note that the system is Markov in the pair $(v_i, l_i)$, so that this step is a straightforward application of the forward-backward algorithm to the graphical model shown in figure 1(d). Observe from this figure that the joint distribution factorizes as:

$$P(V, L, W, X) = P(v_0, l_0) \prod_{i=0}^{|W|-1} P(x_i|v_i) P(\Delta w_i|v_i, l_i) P(v_{i+1}, l_{i+1}|v_i, l_i, \Delta w_i) \qquad (3)$$

From equation 2, (with $B$ instead of $A$), $P(\Delta w_i|v_i, l_i) = B_{v_i}(l_i + \Delta w_i)e^{\left(-\int_{l_i}^{(l_i + \Delta w_i)} B_{v_i}(t)\mathrm{d}t\right)}$.
The term $P(v_{i+1}, l_{i+1}|v_i, l_i, \Delta w_i)$ is the thinning/state-transition probability from steps 4 and 5 of algorithm 1. The forward-filtering stage then moves sequentially through the times in $W$, successively calculating the probabilities $P(v_i, l_i, w_{1:i+1}, x_{1:i})$ using the recursion:

$$P(v_i, l_i, w_{1:i+1}, x_{1:i}) = P(x_i|v_i) P(w_{i+1}|v_i, l_i) \sum_{v_{i-1}, l_{i-1}} P(v_i, l_i|v_{i-1}, l_{i-1}, \Delta w_i) P(v_{i-1}, l_{i-1}, w_{1:i}, x_{1:i-1})$$

The backward sampling stage then returns a new trajectory $(\tilde{V}, \tilde{L}, W) \equiv (\tilde{S}, \tilde{T})$. See figure 1(e).

Observe that $l_i$ can take $(i+1)$ values (in the set $\{0, w_i - w_{i-1}, \cdots, w_i - w_0\}$), with the value of $l_i$ affecting $P(v_{i+1}, l_{i+1}|v_i, l_i, \Delta w_{i+1})$. Thus, the forward-backward algorithm for a general sMJP scales quadratically with $|W|$. We can however use ideas from discrete-time MCMC to reduce this cost (eg. [11] use a slice sampler to limit the maximum holding time of a state, and thus limit $l_i$).

**Resampling the thinned events given the sMJP trajectory:**
Having obtained a new sMJP trajectory $(V, L, W)$, we discard all thinned events $U$, so that the current state of the sampler is now $(S, T)$. We then resample the thinned events $\tilde{U}$, recovering a new thinned representation $(\tilde{V}, \tilde{L}, \tilde{W})$, and with it, a new discretization of time. To simplify notation, we define the *instantaneous* hazard functions $A(t)$ and $B(t)$ (see figure 1(a)):

$$A(t) = A_{\mathbf{S}(t)}(\mathbf{L}(t)), \quad \text{and} \quad B(t) = B_{\mathbf{S}(t)}(\mathbf{L}(t)) \qquad (4)$$

These were the event rates relevant at any time $t$ during the generative process. Note that the sMJP trajectory completely determines these quantities. The events $W$ (whether thinned or not) were generated from a rate $B(\cdot)$ process, while the probability that an event $w_i$ was thinned is $1 - A(w_i)/B(w_i)$. The Poisson thinning theorem [17] then suggests that the thinned events $U$ are distributed as a Poisson process with intensity $(B(t) - A(t))$. The following proposition (see the supplementary material for a proof) shows that this is indeed the case.

**Proposition 2.** *Conditioned on a trajectory $(S, T)$ of the sMJP, the thinned events $U$ are distributed as a Poisson process with intensity $(B(t) - A(t))$.*

Observe that this is independent of the observations $X$. We show in section 2.4 how sampling from such a Poisson process is straightforward for appropriately chosen bounding rates $B_s$.

## 2.3 Related work

An increasingly popular approach to inference in continuous-time systems is particle MCMC (pMCMC) [12]. At a high level, this uses particle filtering to generate a continuous-time trajectory, which then serves as a proposal for a Metropolis-Hastings (MH) algorithm. Particle filtering however cannot propogate back information from future observations, and pMCMC methods can have difficulty in situations where strong observations cause the posterior to deviate from the prior.

Recently, [13] proposed a sampler for MJPs that is a special case of ours. This was derived via a classical idea called uniformization, and constructed the time discretization $W$ from a homogeneous Poisson process. Our sampler reduces to this when a constant dominating rate $B > \max_{s,\tau} A_s(\tau)$ is used to bound all event rates. However, such a 'uniformizing' rate does not always exist (we will discuss two such systems with unbounded rates). Moreover, with a single rate $B$, the average number of candidate events $|W|$, (and thus the computational cost of the algorithm), scales with the leaving rate of the most unstable state. Since this state is often the one that the system will spend the least amount of time in, such a strategy can be wasteful. Under our sampler, the distribution of $W$ is *not* a Poisson process. Instead, events rates are coupled via the sMJP state. This allows our sampler to adapt the granularity of time-discretization to that required by the posterior trajectories, moreover this granularity can vary over the time interval.

There exists other work on continuous-time models based on the idea of a random discretization of time [18, 1]. Like uniformization, these all are limited to specific continuous-time models with specific thinning constructions, and are not formulated in as general a manner as we have done. Moreover, none of these exploit the ability to efficiently resample the time-discretization from a Poisson process, or a new trajectory using the forward-backward algorithm.

## 2.4 Experiments

In this section, we evaluate our sampler on a 3-state sMJP with Weibull hazard rates. Here

$$r_{ss'}(\tau|\alpha_{ss'},\lambda_{ss'}) = e^{(-(\tau/\lambda_{ss'})^{\alpha_{ss'}})} \frac{\alpha_{ss'}}{\lambda_{ss'}} \left(\frac{\tau}{\lambda_{ss'}}\right)^{\alpha_{ss'}-1}, \quad A_{ss'}(\tau|\alpha_{ss'},\lambda_{ss'}) = \frac{\alpha_{ss'}}{\lambda_{ss'}} \left(\frac{\tau}{\lambda_{ss'}}\right)^{\alpha_{ss'}-1}$$

where $\lambda_{ss'}$ is the scale parameter, and the shape parameter $\alpha_{ss'}$ controls the stability of a state $s$. When $\alpha_{ss'} < 1$, on entering state $s$, the system is likely to quickly jump to state $s'$. By contrast, $\alpha_{ss'} > 1$ gives a 'recovery' period before transitions to $s'$. Note that for $\alpha_{ss'} < 1$, the hazard function tends to infinity as $\tau \to 0$. Now, choose an $\Omega > 1$. We use the following simple upper bound $B_{ss'}(\tau)$:

$$B_{ss'}(\tau) = \Omega A_{ss'}(\tau|\alpha_{ss'},\lambda_{ss'}) = \frac{\Omega\alpha_{ss'}}{\lambda_{ss'}} \left(\frac{\tau}{\lambda_{ss'}}\right)^{\alpha_{ss'}-1} = \frac{\alpha_{ss'}}{\tilde{\lambda}_{ss'}} \left(\frac{\tau}{\tilde{\lambda}_{ss'}}\right)^{\alpha_{ss'}-1} \tag{5}$$

Here, $\tilde{\lambda} = \lambda/\sqrt[\alpha]{\Omega}$ for any $\lambda$ and $\alpha$. Thus, sampling from the dominating hazard function $B_{ss'}(\cdot)$ reduces to straightforward sampling from a Weibull with a smaller scale parameter $\tilde{\lambda}_{ss'}$. Note from algorithm 1 that with this construction of the dominating rates, each candidate event is rejected with probability $\left(1 - \frac{1}{\Omega}\right)$; this can be a guide to choosing $\Omega$. In our experiments, we set $\Omega$ equal to 2.

Sampling thinned events on an interval $(t_i, t_{i+1})$ (where the sMJP is in state $s_i$) involves sampling from a Poisson process with intensity $(B(t) - A(t)) = (\Omega - 1)A(t) = (\Omega - 1)\sum_{s'} A_{s_i s'}(t - t_i)$. This is just the superposition of $N$ independent and shifted Poisson processes on $(0, t_{i+1} - t_i)$, the $n$th having intensity $(\Omega - 1)A_{s_i n}(\cdot) \equiv \hat{A}_{s_i n}(\cdot)$. As before, $\hat{A}(\cdot)$ is a Weibull hazard function obtained by correcting the scale parameter $\lambda$ of $A(\cdot)$ by $\sqrt[\alpha]{\Omega - 1}$. A simple way to sample such a Poisson process is by first drawing the number of events from a Poisson distribution with mean $\int_0^{(t_{i+1}-t_i)} \hat{A}_{s_i n}(u)\mathrm{d}u$, and then drawing that many events i.i.d. from $\hat{A}_{s_i n}$ truncated at $(t_{i+1} - t_i)$. Solving the integral for the Poisson mean is straightforward for the Weibull. Call the resulting Poisson sequence $\tilde{T}_n$, and define $\tilde{T} = \cup_{n \in \mathcal{S}} \tilde{T}_n$. Then $W_i \equiv \tilde{T} + t_i$ is the set of resampled thinned events on the interval $(t_i, t_{i+1})$. We repeat this over each segment $(t_i, t_{i+1})$ of the sMJP path.

In the following experiments, the shape parameters for each Weibull hazard ($\alpha_{ss'}$) was randomly drawn from the interval $[0.6, 3]$, while the scale parameter was always set to 1. $\pi_0$ was set to the discrete uniform distribution. The unbounded hazards associated with $\alpha_{ss'} < 1$ meant that uniformization is not applicable to this problem, and we only compared our sampler with pMCMC. We implemented both samplers in Matlab. Our MCMC sampler was set up with $\Omega = 2$, so that the dominating hazard rate at any instant equalled twice the true hazard rate (i.e. $B_{ss'}(\tau) = 2A_{ss'}(\tau)$), giving a probability of thinning equal to 0.5. For pMCMC, we implemented the particle independent Metropolis-Hastings sampler from [12]. We tried different values for the number of particles; for our problems, we found 10 gave best results.

All MCMC runs consisted of 5000 iterations following a burn-in period of 1000. After any MCMC run, given a sequence of piecewise constant trajectories, we calculated the empirical distribution of

Figure 2: ESS per unit time vs the inverse-temperature of the likelihood, when the trajectories are over an interval of length 20 (left) and 2 (right).

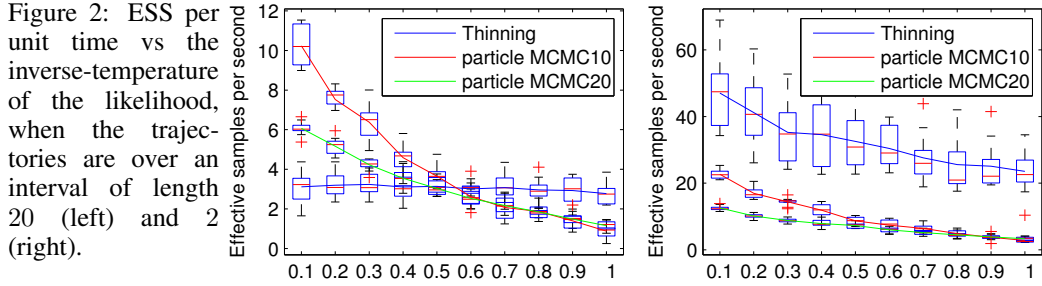

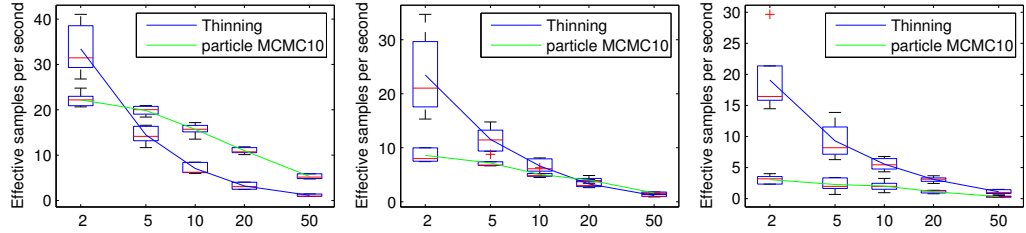

Figure 3: ESS per second for increasing interval lengths. Temperature decreases from the left to right subplots.

the time spent in each state as well as the number of state transitions. We then used R-coda [19] to estimate effective sample sizes (ESS) for these quantities. The ESS of the simulation was set to the median ESS of all these statistics.

**Effect of the observations** For our first experiment, we distributed 10 observations over an interval of length 20. Each observation favoured a particular, random state over the other two states by a factor of 100, giving random likelihood vectors like $(1, 100, 1)^\top$. We then raised the likelihood vector $P(x_i|\cdot)$ to an 'inverse-temperature' $\nu$, so that the effective likelihood at the $i$th observation was $(P(x_i|s_i))^\nu$. As this parameter varied from 0 to 1, the problem moved from sampling from the prior to a situation where the trajectory was observed (almost) perfectly at 10 random times.

The left plot in figure 2 shows the ESS produced per unit time by both samplers as the inverse-temperature increased, averaging results from 10 random parametrizations of the sMJP. We see (as one might expect), that when the effect of the observations is weak, particle MCMC (which uses the prior distribution to make local proposals), outperforms our thinning-based sampler. pMCMC also has the benefit of being simpler implementation-wise, and is about 2-3 times faster (in terms of raw computation time) for a Weibull sMJP, than our sampler. As the effect of the likelihood increases, pMCMC starts to have more and more difficulty tracking the observations. By contrast, our sampler is fairly insensitive to the effect of the likelihood, eventually outperforming the particle MCMC sampler. While there exist techniques to generate more data-driven proposals for the particle MCMC [12, 20], these compromise the appealing simplicity of the original particle MCMC sampler. Moreover, none of these really have the ability to propagate information back from the future (like the forward-backward algorithm), rather they make more and more local moves (for instance, by updating the sMJP trajectory on smaller and smaller subsets of the observation interval).

The right plot in figure 2 shows the ESS per unit time for both samplers, now with the observation interval set to a smaller length of 2. Here, our sampler comprehensively outperforms pMCMC. There are two reasons for this. First, more observations per unit time requires rapid switching between states, a deviation from the prior that particle filtering is unlikely to propose. Additionally, over short intervals, the quadratic cost of the forward-backward step of our algorithm is less pronounced.

**Effect of the observation interval length** In the next experiment, we more carefully compare the two samplers as the interval length varies. For three setting of the inverse temperature parameter (0.1, 0.5 and 0.9), we calculated the number of effective samples produced per unit time as the length of the observation interval increased from 2 to 50. Once again, we averaged results from 10 random settings of the sMJP parameters. Figure 3 show the results for the low, medium and high settings of the the inverse temperature. Again, we clearly see the benefit of the forward-backward algorithm, especially in the low temperature and short interval regimes where the posterior deviates from the prior. Of course, the performance of our sampler can be improved further using ideas from the discrete-time domain; these can help ameliorate effect of the quadratic cost for long intervals.

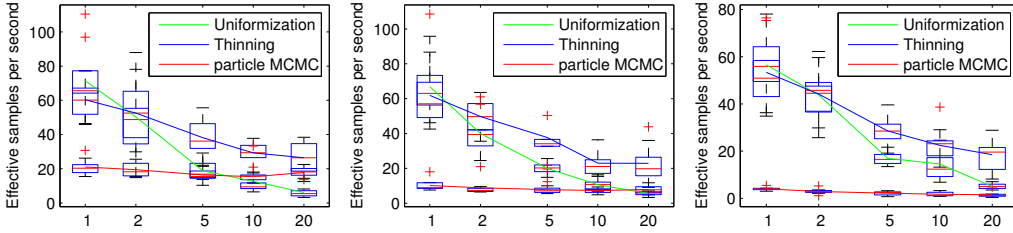

Figure 4: Effect of increasing the leaving rate of a state. Temperature decreases from the left to right plots.

## 3  Markov jump processes

In this section, we look at the Markov jump process (MJP), which we saw has constant hazard functions $A_{ss'}$. MJPs are also defined to disallow self-transitions, so that $A_{ss} = 0 \; \forall s \in \mathcal{S}$. If we use constant dominating hazard rates $B_s$, we see from algorithm 1 that all probabilities at time $w_i$ depend only on the current state $s_i$, and are independent of the holding time $l_i$. Thus, we no longer need to represent the holding times $L$. The forward message at time $w_i$ needs only to represent the probability of $v_i$ taking different values in $\mathcal{S}$; this completely specifies the state of the MJP. As a result, the cost of a forward-backward iteration is now linear in $|W|$.

In the next experiment, we compare Matlab implementations of our thinning-based sampler and the particle MCMC sampler with the uniformization-based sampler described in section 2.3. Recall that the latter samples candidate event times $W$ from a homogeneous Poisson process with a state-independent rate $B > \max_s A_s$. Following [13], we set $B = 2 \max_s A_s$. As in section 2.4, we set $\Omega = 2$ for our sampler, so that $B_s = 2A_s \; \forall s$. pMCMC was run with 20 particles.

Observe that for uniformization, the rate $B$ is determined by the leaving rate of the most unstable state; often this is the state the system spends the least time in. To study this, we applied all three samplers to a 3-state MJP, two of whose states had leaving rates equal to 1. The leaving rate of the third state, was varied from 1 to 20 (call this rate $\gamma$). On leaving any state, the probability of transitioning to either of the other two was uniformly distributed between 0 and 1. This way, we constructed 10 random MJPs for each $\gamma$. We distributed 5 observation times (again, favouring a random state by a factor of 100) over the interval $[0, 10]$. Like section 2.4, we looked at the ESS per unit time for 3 settings of the inverse temperature parameter $\nu$, now as we varied $\gamma$.

Figure 4 shows the results. The pMCMC sampler clearly performs worse than the other two. The Markov structure of the MJP makes the forward-backward algorithm very natural and efficient, by contrast, running a particle filter with 20 particles took about twice as long as our sampler. Further, we see that while both the uniformization and our sampler perform comparably for low values of $\gamma$, our sampler starts to outperform uniformization for $\gamma$'s greater than 2. In fact, for weak observations and large $\gamma$s, even particle MCMC outperforms uniformization. As we mentioned earlier, this is because for uniformization, the granularity of time-discretization is determined by the least stable state, resulting in very long Markov chains for large values of $\gamma$.

### 3.1  The M/M/∞ queue

We finally apply our ideas to an infinite state MJP from queuing theory, the M/M/∞ queue (also called an *immigration-death* process [21]). Here, individuals (customers, messages, jobs etc.) enter a population according to a homogeneous Poisson process with rate $\alpha$ independent of the population size. The lifespan of each individual (or the job 'service time') is exponentially distributed with rate $\beta$, so that the rate at which a 'death' occurs in the population is proportional to the population size.

Let $\mathbf{S}(t)$ represent the population size (or the number of 'busy servers') at time $t$. Then, under the M/M/∞ queue, the stochastic process $\mathbf{S}(t)$ evolves according to a simple birth-death Markov jump process on the space $\mathcal{S} = \{1, \cdots, \infty\}$, with rates $A_{s,s+1} = \alpha$ and $A_{s,s-1} = s\beta$. All other rates are 0. Observe that since the population size of the M/M/∞ queue is unbounded, we cannot upper bound the event rates in the system. Thus, uniformization is not directly applicable to this system. Instead, we have to truncate the maximum value of $\mathbf{S}(t)$ to some constant, say $c$. This is the so-called M/M/c/c queue; now, when all $c$ servers are busy, any incoming jobs are rejected.

In the following, we considered an M/M/∞ queue with $\alpha$ and $\beta$ set to 10 and 1 respectively. For some $t_{end}$, the state of the system was observed perfectly at three times 0, $t_{end}/10$ and $t_{end}$, with values 10, 2 and 15 respectively. Conditioned on these, we sought the posterior distribution over the

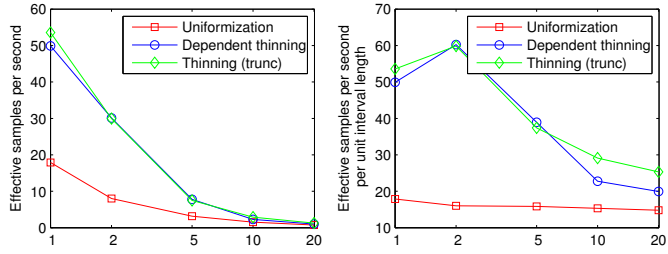

Figure 5: The M/M/$\infty$ queue: a) ESS per unit time b) ESS per unit time scaled by interval length.

system trajectory on the interval $[0, t_{end}]$. Since the state of the system at time 0 is perfectly observed to be 10, given any time-discretization, the maximum value of $s_i$ at step $i$ of the Markov chain is $(10 + i)$. Thus, message dimensions are always finite, and we can directly apply the forward-backward algorithm. For noisy observations, we can use a slice sampler [22]. We compared our sampler with uniformization; for this, we approximated the M/M/$\infty$ system with an M/M/50/50 system. We also applied our sampler to this truncated approximation, labelling it as 'Thinning (trunc)'. For both these samplers, the message dimensions were 50. The large state spaces involved makes pMCMC very inefficient, and we did not include it in our results.

Figure 5(a) shows the ESS per unit time for all three samplers as we varied the interval length $t_{end}$ from 1 to 20. Sampling a trajectory over a long interval will take more time than over a short one, and to more clearly distinguish performance for large values of $t_{end}$, we scale each ESS from the left plot with $t_{end}$, the length of the interval, in the right subplot of figure 5.

We see our sampler always outperforms uniformization, with the difference particularly significant for short intervals. Interestingly, running our thinning-based sampler on the truncated system offers no significant computational benefit over running it on the full model. As the observation interval becomes longer and longer, the MJP trajectory can make larger and larger excursions (especially over the interval $[t_{end}/10, t_{end}]$). Thus as $t_{end}$ increases, event rates witnessed in posterior trajectories starts to increase. As our sampler adapts to this, the number of thinned events in all three samplers start to become comparable, causing the uniformization-based sampler to approach the performance of the other two samplers. At the same time, we see that the difference between our truncated and our untruncated sampler starts to widen. Of course, we should remember that over long intervals, truncating the system size to 50 becomes more likely to introduce biases into our inferences.

## 4  Discussion

We described a general framework for MCMC inference in continuous-time discrete-state systems. Each MCMC iteration first samples a random discretization of time given the trajectory of the system. Given this, we then resample the sMJP trajectory using the forward-backward algorithm. While we looked only at semi-Markov and Markov jump processes, it is easy to extend our approach to piecewise-constant stochastic processes with more complicated dependency structures.

For our sampler, a bottleneck in the rate of mixing is that the new and old trajectories share an intermediate discretization $W$ (see figure 1(e)). Recall that an sMJP trajectory defines an instantaneous hazard function $B(t)$; our scheme requires the discretization sampled from the old hazard function be compatible with the new hazard function. Thus, the forward-backward algorithm is unlikely to return a trajectory associated with a hazard function that differ significantly from the old one. By contrast, for uniformization, the hazard function is a constant $B$, independent of the system state. However, this comes at the cost of a conservatively high discretization of time. An interesting direction for future work is too see how different choices of the dominating hazard function can help trade-off these factors. For instance, we proposed, using a single $\Omega$, with $B_s(\cdot) = \Omega A_s(\cdot)$. It is possible to use a different $\Omega_s$ for each state $s$, or even an $\Omega_s(\cdot)$ that varies with time. Similarly, one can consider additive (rather than multiplicative) constructions of $B_s(\cdot)$.

For general sMJPs, the forward-backward algorithm scales quadratically with $|W|$, the number of candidate jump times. Such scaling is characteristic of sMJPs, though we can avail of discrete-time MCMC techniques to ameliorate this. For sMJPs whose hazard functions are constant beyond a 'window of memory', inference scales quadratically with the memory length, and only linearly with $|W|$. One can use such approximations to devise efficient MH proposals for sMJPs trajectories.

# References

[1] Ryan P. Adams, Iain Murray, and David J. C. MacKay. Tractable nonparametric Bayesian inference in Poisson processes with Gaussian process intensities. In *Proceedings of the 26th International Conference on Machine Learning (ICML)*, 2009.

[2] Y. W. Teh, C. Blundell, and L. T. Elliott. Modelling genetic variations with fragmentation-coagulation processes. In *Advances In Neural Information Processing Systems*, 2011.

[3] U. Nodelman, C.R. Shelton, and D. Koller. Continuous time Bayesian networks. In *Proceedings of the Eighteenth Conference on Uncertainty in Artificial Intelligence (UAI)*, pages 378–387, 2002.

[4] Ardavan Saeedi and Alexandre Bouchard-Côté. Priors over Recurrent Continuous Time Processes. In *Advances in Neural Information Processing Systems 24 (NIPS)*, volume 24, 2011.

[5] Matthias Hoffman, Hendrik Kueck, Nando de Freitas, and Arnaud Doucet. New inference strategies for solving Markov decision processes using reversible jump MCMC. In *Proceedings of the Twenty-Fifth Conference Annual Conference on Uncertainty in Artificial Intelligence (UAI-09)*, pages 223–231, Corvallis, Oregon, 2009. AUAI Press.

[6] A. Doucet, N. de Freitas, and N. J. Gordon. *Sequential Monte Carlo Methods in Practice*. Statistics for Engineering and Information Science. New York: Springer-Verlag, May 2001.

[7] Früwirth-Schnatter. Data augmentation and dynamic linear models. *J. Time Ser. Anal.*, 15:183–202, 1994.

[8] C. K. Carter and R. Kohn. Markov chain Monte Carlo in conditionally Gaussian state space models. *Biometrika*, 83:589–601, 1996.

[9] Radford M. Neal, Matthew J. Beal, and Sam T. Roweis. Inferring state sequences for non-linear systems with embedded hidden Markov models. In *Advances in Neural Information Processing Systems 16 (NIPS)*, volume 16, pages 401–408. MIT Press, 2004.

[10] J. Van Gael, Y. Saatci, Y. W. Teh, and Z. Ghahramani. Beam sampling for the infinite hidden Markov model. In *Proceedings of the International Conference on Machine Learning*, volume 25, 2008.

[11] M. Dewar, C. Wiggins, and F. Wood. Inference in hidden Markov models with explicit state duration distributions. *IEEE Signal Processing Letters*, page To Appear, 2012.

[12] Christophe Andrieu, Arnaud Doucet, and Roman Holenstein. Particle Markov chain Monte Carlo methods. *Journal of the Royal Statistical Society Series B*, 72(3):269–342, 2010.

[13] V. Rao and Y. W. Teh. Fast MCMC sampling for Markov jump processes and continuous time Bayesian networks. In *Proceedings of the International Conference on Uncertainty in Artificial Intelligence*, 2011.

[14] William Feller. On semi-Markov processes. *Proceedings of the National Academy of Sciences of the United States of America*, 51(4):pp. 653–659, 1964.

[15] D. Sonderman. Comparing semi-Markov processes. *Mathematics of Operations Research*, 5(1):110–119, 1980.

[16] D. J. Daley and D. Vere-Jones. *An Introduction to the Theory of Point Processes*. Springer, 2008.

[17] J. F. C. Kingman. *Poisson processes*, volume 3 of *Oxford Studies in Probability*. The Clarendon Press Oxford University Press, New York, 1993. Oxford Science Publications.

[18] A. Beskos and G.O. Roberts. Exact simulation of diffusions. *Annals of applied probability*, 15(4):2422 – 2444, November 2005.

[19] Martyn Plummer, Nicky Best, Kate Cowles, and Karen Vines. CODA: Convergence diagnosis and output analysis for MCMC. *R News*, 6(1):7–11, March 2006.

[20] Andrew Golightly and Darren J. Wilkinson. Bayesian parameter inference for stochastic biochemical network models using particle Markov chain Monte Carlo. *Interface Focus*, 1(6):807–820, December 2011.

[21] S. Asmussen. *Applied Probability and Queues*. Applications of Mathematics. Springer, 2003.

[22] Stephen G. Walker. Sampling the Dirichlet mixture model with slices. *Communications in Statistics - Simulation and Computation*, 36:45, 2007.

